# Learning Exact Patterns of Quasi-synchronization among Spiking Neurons from Data on Multi-unit Recordings

**Laura Martignon**
Max Planck Institute
for Psychological Research
Adaptive Behavior and Cognition
80802 Munich, Germany
laura@mpipf-muenchen.mpg.de

**Kathryn Laskey**
Dept. of Systems Engineering
and the Krasnow Institute
George Mason University
Fairfax, Va. 22030
klaskey@gmu.edu

**Gustavo Deco**
Siemens AG
Central Research
Otto Hahn Ring 6
81730 Munich
gustavo.deco@zfe.siemens.de

**Eilon Vaadia**
Dept. of Physiology
Hadassah Medical School
Hebrew University of Jerusalem
Jerusalem 91010, Israel
eilon@hbf.huji.ac.il

## Abstract

This paper develops arguments for a family of temporal log-linear models to represent spatio-temporal correlations among the spiking events in a group of neurons. The models can represent not just pairwise correlations but also correlations of higher order. Methods are discussed for inferring the existence or absence of correlations and estimating their strength.

A frequentist and a Bayesian approach to correlation detection are compared. The frequentist method is based on $G^2$ statistic with estimates obtained via the Max-Ent principle. In the Bayesian approach a Markov Chain Monte Carlo Model Composition ($MC^3$) algorithm is applied to search over connectivity structures and Laplace's method is used to approximate their posterior probability. Performance of the methods was tested on synthetic data. The methods were applied to experimental data obtained by the fourth author by means of measurements carried out on behaving Rhesus monkeys at the Hadassah Medical School of the Hebrew University. As conjectured, neural connectivity structures need not be neither hierarchical nor decomposable.

# 1 INTRODUCTION

Hebb conjectured that information processing in the brain is achieved through the collective action of groups of neurons, which he called *cell assemblies* (Hebb, 1949). His followers were left with a twofold challenge:

- to define cell assemblies in an unambiguous way.
- to conceive and carry out the experiments that demonstrate their existence.

Cell assemblies have been defined in various sometimes conflicting ways, both in terms of anatomy and of shared function. One persistent approach characterizes the cell assembly by near-simultaneity or some other specific timing relation in the firing of the involved neurons. If two neurons converge on a third one, their synaptic influence is much larger for near-coincident firing, due to the spatio-temporal summation in the dendrite (Abeles,1991; Abeles et al. 1993). Thus *syn-firing* is directly available to the brain as a potential code.

The second challenge has led physiologists to develop methods to observe the simultaneous activity of individual neurons to seek evidence for spatio-temporal patterns. It is now possible to obtain multi-unit recordings of up to 100 neurons in awake behaving animals. In the data we analyze, the spiking events (in the 1 msec range) are encoded as sequences of 0's and 1's, and the activity of the whole group is described as a sequence of binary configurations. This paper presents a statistical model in which the parameters represent spatio-temporal firing patterns. We discuss methods for estimating these pararameters and drawing inferences about which interactions are present.

# 2 PARAMETERS FOR SPATIO-TEMPORAL FIRING PATTERNS

The term spatial correlation has been used to denote synchronous firing of a group of neurons, while the term temporal correlation has been used to indicate chains of firing events at specific temporal intervals. Terms like "couple" or "triplet" have been used to denote spatio-temporal patterns of two or three neurons (Abeles et al., 1993; Grün, 1996) firing simultaneously or in sequence. Establishing the presence of such patterns is not straightforward. For example, three neurons may fire together *more often than expected by chance*[1] without exhibiting an authentic third order interaction. This phenomenon may be due, for instance, to synchronous firing of two couples out of the three neurons. Authentic triplets, and, in general, authentic n-th order correlations, must therefore be distinguished from correlations that can be explained in terms of lower order interactions. In what follows, we present a parameterized model that represents a spatio-temporal correlation by a parameter that depends on the involved neurons and on a set of time intervals, where synchronization is characterized by all time intervals being zero.

Assume that the sequence of configurations $\underline{x}_t = (x_{(1t)}, \cdots, x_{(Nt)})$ of $N$ neurons forms a Markov chain of order $r$. Let $\delta$ be the time step, and denote the conditional distribution for $\underline{x}_t$ given previous configurations by $p(\underline{x}_t \mid \underline{x}_{(t-\delta)}, \underline{x}_{(t-2\delta)}, \ldots, \underline{x}_{(t-r\delta)})$. We assume that all transition probabilities are strictly positive and expand the logarithm of the conditional distribution as:

$$p(\underline{x}_t \mid \underline{x}_{(t-\delta)}, \underline{x}_{(t-2\delta)}, ..., \underline{x}_{(t-r\delta)}) = exp\{\theta_0 + \sum_{A \in \Xi} \theta_A X_A\} \quad (1)$$

where each $A$ is a subset of pairs of subscripts of the form $(i, t - s\delta)$ that includes at least one pair of the form $(i,t)$. Here $X_A = \prod_{1 \le j \le k} x_{(i_j, t-m_j\delta)}$ denotes the event that all neurons in $A$ are active. The set $\Xi \subseteq 2^A$ of all subsets for which $\theta_A$ is non-zero is called the *interaction structure* for the distribution $p$. The effect $\theta_A$ is called the *interaction strength* for the interaction on subset $A$. Clearly, $\theta_A = 0$ is equivalent to $A \notin \Xi$ and is taken to indicate absence of an order-$|A|$ interaction among neurons in $A$. We denote the structure-specific vector of non-zero interaction strengths by $\underline{\theta}_\Xi$. Consider a set $\Lambda$ of $N$ binary neurons and denote by $p$ the probability distribution on the binary configurations of $\Lambda$.

**DEFINITION 1:** We say that neurons $\{i_1, i_2, ..., i_k\}$ exhibit a *spatio-temporal pattern* if there is a set of time intervals $m_1\delta, m_2\delta, ..., m_k\delta$ with at least one $m_i = 0$, such that $\theta_A \neq 0$ in Equation (1), where
$A = \{(i_1, t - m_1\delta), ..., (i_k, t - m_k\delta)\}$.

**DEFINITION 2:** A subset $\{i_1, i_2, ..., i_k\}$ of neurons exhibits a *synchronization* or *spatial correlation* if $\theta_A \neq 0$ for $A = \{(i_1, 0), ..., (i_k, 0)\}$.

In the case of absence of any temporal dependencies the configurations are independent and we drop the time index:

$$p(\underline{x}) = exp\{\theta_0 + \sum \theta_A X_A\} \quad (2)$$

where A is any nonempty subset of $\Lambda$ and $X_A = \prod_{i \in A} x_i$.

Of course (2) is unrealistic. Temporal correlation of some kind is always present, one such example being the refractory period after firing. Nevertheless, (2) may be adequate in cases of weak temporal correlation. Although the models (1) and (2) are statistical not physiological, it is an established conjecture that synaptic connection between two neurons will manifest as a non-zero $\theta_A$ for the corresponding set $A$ in the temporal model (1). Another example leading to non-zero $\theta_A$ will be simultaneous activation of the neurons in $A$ due to a common input, as illustrated in Figure 1 below. Such a $\theta_A$ will appear in model (1) with time intervals equal to 0. An attractive feature of our models is that it is capable of distinguishing between cases a. and b. of Figure 1. This can be seen by extending the model (2) to include the external neurons (H in case a., H,K in case b.) and then marginalizing. An information-theoretic argument supports the choice of $\theta_A \neq 0$ as a natural indicator of an order-$|A|$ interaction among the neurons in $A$. Assume that we are in the case of no temporal correlation. The absence of interaction of order $|A|$

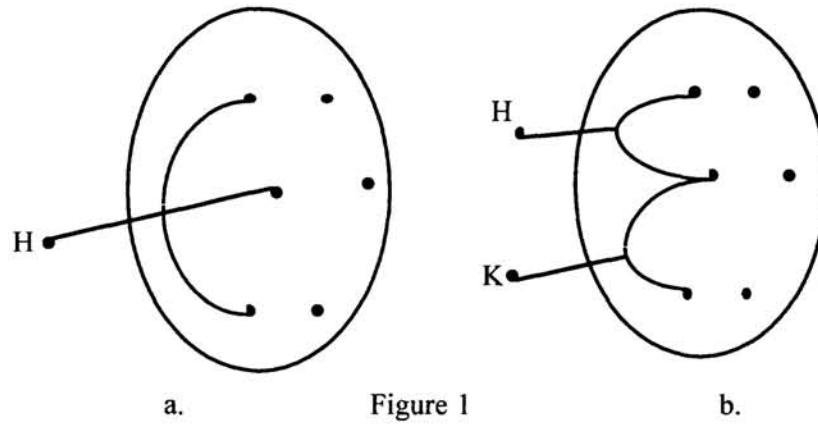

a.                    Figure 1                    b.

among neurons in $A$ should be taken to mean that the distribution is determined by the marginal distributions on proper subsets of $A$. A well established criterion for selecting a distribution among those matching the lower order marginals fixed by proper subsets of $A$, is Max-Ent. According to the Max-Ent principle the distribution that maximizes entropy is the one which is maximally non-committal with regard to missing information. The probability distribution $p^*$ that maximizes entropy among distributions *with the same marginals as the distribution* $p$ *on proper subsets of* $A$ has a log-linear expansion in which only $\theta_B, B \subset A, B \neq A$ can possibly be non-zero.[2]

## 3 THE FREQUENTIST APPROACH

We treat here the case of no temporal dependencies. The general case is treated in Martignon-Deco,1997; Deco-Martignon,1997. We also assume that our data are stationary. We test the presence of synchronization of neurons in $A$ by the following procedure: we condition on silence of neurons in the complement of $A$ in $\Lambda$ and call the resulting frequency distribution $p$. We construct the Max-Ent model determined by the marginals of $p$ on proper subsets of $A$. The well-known method for constructing this type of Max-Ent models is the I.P.F.P. Algorithm (Bishop et al.,1975). We propose here another simpler and quicker procedure:

If $B$ is a subset of $A$, denote by $\chi_B$ the configuration that has a component $1$ for every index in $B$ and $0$ elsewhere.

Define $p^*(\chi_B) = p(\chi_B) + (-1)^{|B|}\Delta$, where $\Delta$ is to be determined by solving for $\theta^*_A \equiv 0$, where $\theta^*_A$ is the coefficient corresponding to $A$ in the log-expansion of $p^*$. As can be shown (Martignon et al, 1995), $\theta^*_A$ can be written as

$\theta^*_A = \sum_{B \subset A} (-1)^{|A-B|} \ln p^*(\chi_B)$. The distribution $p^*$ maximizes entropy among those with the same marginals of $p$ on proper subsets of $A$.[3] We use $p^*$ as estimate of $p$ for tests by means of $G^2$ statistic (Bishop et al., 1975).

## 4  THE BAYESIAN APPROACH

We treat here the case of no temporal dependencies. The general case is treated in Laskey-Martignon, 1997. Information about $p(x)$ prior to observing any data is represented by a joint probability distribution called the *prior distribution* over $\Xi$ and the $\theta$'s. Observations are used to update this probability distribution to obtain a *posterior distribution* over structures and parameters. The posterior probability of a cluster $A$ can be interpreted as the probability that the $r$ nodes in cluster $A$ exhibit a degree-$r$ interaction. The posterior distribution for $\theta_A$ represents structure-specific information about the magnitude of the interaction. The mean or mode of the posterior distribution can be used as a point estimate of the interaction strength; the standard deviation of the posterior distribution reflects remaining uncertainty about the interaction strength.

We exhibit a family of log-linear models capable of capturing interactions of all orders. An algorithm is presented for learning both structure and parameters in a unified Bayesian framework. Each model structure specifies a set of clusters of nodes, and structure-specific parameters represent the directions and strengths of interactions among them. The Bayesian learning algorithm gives high posterior probability to models that are consistent with the data. Results include a probability, given the observations, that a set of neurons fires simultaneously, and a posterior probability distribution for the strength of the interaction, conditional on its occurrence.

The prior distribution we used has two components. The first component assigns a prior probability to each structure. In our model, interactions are independent of each other and each interaction has a probability of .1. This reflects the prior expectation that not many interactions are expected to be present. The second component of the prior distribution is the conditional distribution of interaction strengths given the structure. If an interaction is not in the structure, the corresponding strength parameter $\theta_A$ is taken to be identically zero given structure $\Xi$. All interactions belonging to $\Xi$ are taken to be independent and normally distributed with mean zero and standard deviation 2. This reflects the prior expectation that interaction strength magnitudes are rarely larger than 4 in absolute value.

Computing the posterior probability of a structure $\Xi$ requires integrating out of the joint mass-density function of the structure $\Xi$, the interaction strength $\theta_A$, and the data $X$. The solution to this integral cannot be obtained in closed form. We use Laplace's method (Kass-Raftery, 1995; Tierney-Kadane,1986) to estimate the posterior probability of structures. The posterior distribution of $\theta_A$ given frequency data also

cannot be obtained in closed form. We use the mode of the posterior distribution as a point estimate of $\theta_A$. The standard deviation of $\theta_A$, which indicates how precisely $\theta_A$ can be estimated from the given data, is estimated using a normal approximation to the posterior distribution (Laskey-Martignon, 1997). The covariance matrix of the $\theta_A$ is estimated as the inverse Fisher information matrix evaluated at the mode of the posterior distribution. The posterior probability of an interaction $\theta_A$ is the sum over the posterior probabilities of all structures containing $A$. We used a Markov chain Monte Carlo Model Composition algorithm ($MC^3$) to search over structures. This stochastic algorithm converges to a stationary distribution in which structure $\Xi$ is visited with probability equal to its posterior probability. We ran the $MC^3$ algorithm for 15,000 runs and estimated the posterior probability of a structure as its frequency of occurrence over the 15,000 runs. We estimated interaction strength parameters and standard deviations using only the 100 highest-probability structures. Although the number of possible structures is astronomical, typically most of the posterior probability is contained in relatively few structures. We found this to be the case, which justifies using only the most probable structures to estimate interaction strength parameters.

## 5   RESULTS

We applied our models to data from an experiment in which spiking events among groups of neurons were analyzed through multi-unit recordings of 6-16 units in the frontal cortex of Rhesus monkeys. The monkeys were trained to localize a source of light and, after a delay, to touch the target from which the light blink was presented. At the beginning of each trial the monkeys touched a "ready-key", then the central ready light was turned on. Later, a visual cue was given in the form of a 200-ms light blink coming from either the left or the right. Then, after a delay of 1 to 32 seconds, the color of the ready light changed from red to orange and the monkeys had to release the ready key and touch the target from which the cue was given. The spiking events (in the 1 millisecond range) of each neuron were encoded as a sequence of zeros and ones, and the activity of the group was described as a sequence of configurations of these binary states. The fourth author provided data corresponding to piecewise stationary segments of the trials, which presented weak temporal correlation, corresponding to intervals of 2000 milliseconds around the ready-signal. He adjoined these 94 segments and formed a data-set of 188,000 msec. The data were then binned in time windows of 40 milliseconds. The criterion we used to fix the binwidth was robustness with regards to variations of the offsets. We selected a subset of eight of the neurons for which data were recorded. We analyzed recordings prior to the ready-signal separately from data recorded after the ready-signal. Each of these data sets is assumed to consist of independent trials from a model of the form (2).

| Cluster A | Posterior prob. of A (frequency) | Posterior prob. of A (best 100models) | MAP estimate of $\theta_A$ | Standard deviation of $\theta_A$ | Significance |
|---|---|---|---|---|---|
| 6,8 | .9 | .89 | 0.47 | 0.11 | 4.0853 |
| 4,5,6,7 | .30 | 0.32 | 2.30 | 0.64 | No |
| 2,3,6 | .40 | 0.38 | 2.30 | 0.64 | 2.35 |
| 1,3,4 | close to prior | close to prior | | | 4.7 |

Table1: results for pre-ready signal data. Effects with posterior prob. > 0.1

| Cluster A | Posterior prob. of A (frequency) | Posterior prob. of A (best 100 models) | MAP estimate of $\theta_A$ | Standard deviation of $\theta_A$ | Significance |
|---|---|---|---|---|---|
| 5,6 | .79 | 0.96 | 1.00 | 0.27 | 1.82 |
| 4,7 | .246 | 0.18 | 0.93 | 0.34 | 2.68 |
| 1,4,5,6 | 0.18 | 0.13 | 1.06 | 0.36 | No |
| 1,3,4,6,7 | 0.24 | 0.17 | 2.69 | 0.13 | No |

Table2:results for post-ready signal data.   Effects with posterior prob >0.1

Another set of data from 5 simulated neurons  was provided by the fourth author for a double-check of the methods.   Only second order correlations had been simulated: a synapse lasting 2 msec, an inhibitory common input, and two excitatory common inputs. The Bayesian method was very accurate, detecting exactly the simulated interactions. The frequentist method made one mistake.   Other data sets with temporal correlations have also been analyzed.  By means of the frequentist approach on shifted data, temporal triplets have been detected and even fourth order correlations.  Temporal correlograms are computed for shifts of up to 50 msec (Martignon-Deco, 1997).

## Footnotes

[1] that is to say, more often than predicted by the null hypothesis of independence.

[2] This was observed by J. Good in 1963 (Bishop et al. 1975). It is interesting to note that $p^*$ minimizes the Kullback-Leibler distance from $p$ in the manifold of distributions with a log-linear expansion in which only $\theta_B, B \subset A, B \neq A$ can possibly be non-zero.

[3] This is due to the fact that there is a unique distribution with the same marginals of $p$ on proper subsets of $A$ such that the coefficient corresponding to $A$ in its log-expansion is zero.

## References

Hebb, D.  (1949) *The Organization of Behavior.*  New York: Wiley, 1949.

Abeles, M.(1991)*Corticonics: Neural Circuits of the Cerebral Cortex.*  Cambridge: Cambridge University Press, 1991.

Abeles, M., H. Bergman, E. Margalit, and E. Vaadia.  (1993) "Spatiotemporal Firing Patterns in the Frontal Cortex of Behaving Monkeys." *Journal of Neurophysiology*  70,  4:,  1629-1638.

Grün S. (1996)  *Unitary Joint-Events in Multiple-Neuron Spiking Activity-Detection, Significance and Interpretation.*  Verlag Harry Deutsch, Frankfurt.

Martignon L.  and Deco G.  (1997) "Neurostatistics of Spatio-Temporal Patterns of Neural Activation: the frequentist approach" Technical Report, MPI-ABC no.*3.*

Deco G.  and Martignon L.  (1997) "Higher-order Phenomena among Spiking Events of Groups of Neurons" Preprint.

Bishop, Y., S. Fienberg, and P. Holland (1975) *Discrete Multivariate Analysis.*  Cambridge, MA: MIT Press.

Martignon L.,.v.Hasseln H.  Grün S, Aertsen A, Palm G.(1995) "Detecting Higher Order Interactions among the Spiking Events of a Group of Neurons" Biol.Cyb. *73*,  69-81.

Kass, .  and Raftery A.  (1995) "Bayes factors"*Journal of the American Statistical Association 90,* no. *430:,*  773-795.

Tierney, L., and J. B. Kadane (1986) "Accurate Approximations for Posterior Moments and Marginal Densities." *Journal of the American Statistical Association 81*,  82-86

Laskey K., and Martignon L.(1997) "Neurostatistics of Spatio-temporal Patterns of Neural Activation: the Bayesian Approach", in preparation

Laskey K., and  Martignon, L.(1996) "Bayesian Learning of Log-linear Models for Neural Connectivity" *Proceedings of the XII Conference on Uncertainty in Artificial Intelligence*, Horvitz E. ed., Morgan-Kaufmann, San Mateo.
